# Lipreading by neural networks: Visual preprocessing, learning and sensory integration

**Gregory J. Wolff**
Ricoh California Research Center
2882 Sand Hill Road Suite 115
Menlo Park, CA 94025-7022
wolff@crc.ricoh.com

**K. Venkatesh Prasad**
Ricoh California Research Center
2882 Sand Hill Road Suite 115
Menlo Park, CA 94025-7022
prasad@crc.ricoh.com

**David G. Stork**
Ricoh California Research Center
2882 Sand Hill Road Suite 115
Menlo Park, CA 94025-7022
stork@crc.ricoh.com

**Marcus Hennecke**
Department of Electrical Engineering
Mail Code 4055
Stanford University
Stanford, CA 94305

## Abstract

We have developed visual preprocessing algorithms for extracting phonologically relevant features from the grayscale video image of a speaker, to provide speaker-independent inputs for an automatic lipreading ("speechreading") system. Visual features such as mouth open/closed, tongue visible/not-visible, teeth visible/not-visible, and several shape descriptors of the mouth and its motion are all rapidly computable in a manner quite insensitive to lighting conditions. We formed a hybrid speechreading system consisting of two time delay neural networks (video and acoustic) and integrated their responses by means of independent opinion pooling — the Bayesian optimal method given conditional independence, which seems to hold for our data. This hybrid system had an error rate 25% lower than that of the acoustic subsystem alone on a five-utterance speaker-independent task, indicating that video can be used to improve speech recognition.

# 1   INTRODUCTION

Automated speech recognition is notoriously hard, and thus any predictive source of information and constraints that could be incorporated into a computer speech recognition system would be desirable. Humans, especially the hearing impaired, can utilize visual information — "speech reading" — for improved accuracy (Dodd & Campbell, 1987, Sanders & Goodrich, 1971). Speech reading can provide direct information about segments, phonemes, rate, speaker gender and identity, and subtle information for segmenting speech from background noise or multiple speakers (De Filippo & Sims, 1988, Green & Miller, 1985).

Fundamental support for the use of visual information comes from the complementary nature of the visual and acoustic speech signals. Utterances that are difficult to distinguish *acoustically* are the easiest to distinguish *visually*, and vice versa. Thus, for example /mi/ ↔ /ni/ are highly confusable acoustically but are easily distinguished based on the visual information of lip closure. Conversely, /bi/ ↔ /pi/ are highly confusable visually ("visemes"), but are easily distinguished acoustically by the voice-onset time (the delay between the burst sound and the onset of vocal fold vibration). Thus automatic lipreading promises to help acoustic speech recognition systems for those utterances where they need it most; visual information cannot contribute much information to those utterances that are already well recognized acoustically.

## 1.1   PREVIOUS SYSTEMS

The system described below differs from recent speech reading systems. Whereas Yuhas et al. (1989) recognized *static* images and acoustic spectra for vowel recognition, ours recognizes *dynamic* consonant-vowel (CV) utterances. Whereas Petajan, Bischoff & Bodoff (1988) used thresholded pixel based representations of speakers, our system uses more sophisticated visual preprocessing to obtain phonologically relevant features. Whereas Pentland and Mase (1989) used optical flow methods for estimating the motion of four lip regions (and used no acoustic subsystem), we obtain several other features from intensity profiles. Whereas Bregler et al. (1993) used direct pixel images, our recognition engine used a far more compressed visual representation; our method of integration, too, was based on statistical properties of our data. We build upon the basic recognizer architecture of Stork, Wolff and Levine (1992), but extend it to grayscale video input.

# 2   VISUAL PREPROCESSING

The sheer quantity of image data presents a hurdle to utilizing video information for speech recognition. Our approach to video preprocessing makes use of several simple computations to reduce the large amount of data to a manageable set of low-level image statistics describing the region of interest around the mouth. These statistics capture such features as the positions of the upper and lower lip, the mouth shape, and their time derivatives. The rest of this section describes the computation of these features.

Grayscale video images are captured at 30 frames/second with a standard NTSC

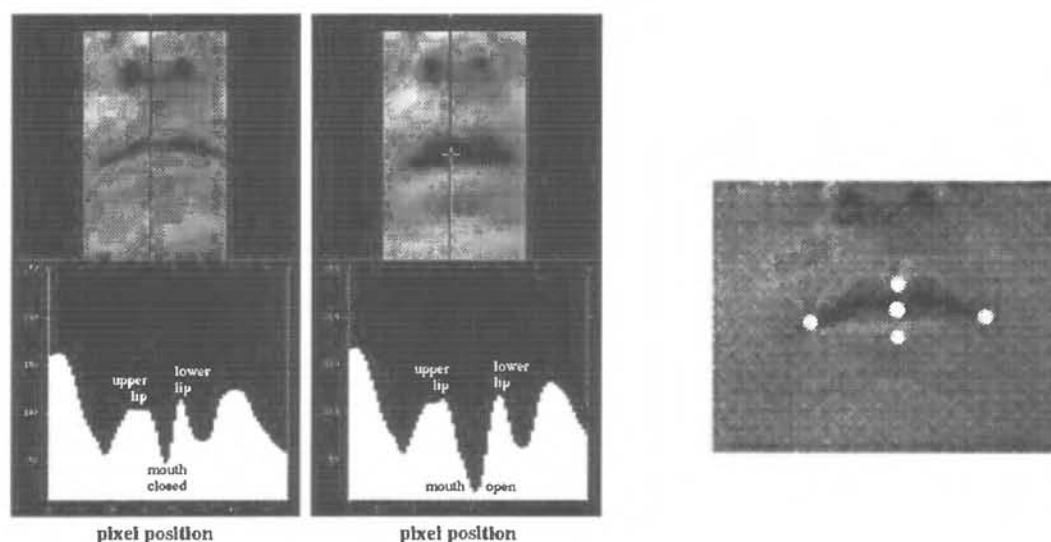

Figure 1: (Left) The central bands of the automatically determined ROI from two frames of the video sequence of the utterance /ba/ and their associated luminance profiles along the central marked line. Notice that the lowest valley in this profile changes drastically in intensity as the mouth changes from closed to open. In addition, the linear separation between the peaks adjacent to the lowest valley also increases as the mouth opens. These features are identified on the ROI from a single frame (right). The position, intensity, and temporal variation of these features provide input to our recognizer.

camera, and subsampled to give 150 x 150 pixel image sequence. A 64 × 64 pixel region of interest (ROI) is detected and tracked by means of the following operations on the full video images:

- Convolve with 3 × 3 pixel low-pass filter      (to remove spatial noise)
- Convolve with 3 × 3 pixel edge detector        (to detect edges)
- Convolve with 3 × 3 pixel low-pass filter      (to smooth edges)
- Threshold at $(I_{max} - I_{min})/2$            (to isolate eyes and mouth)
- Triangulate eyes with mouth                     (to obtain ROI)

We also use temporal coherence in frame-to-frame correlations to reduce the effects of noise in the profile or missing data (such as "closed" eyes). Within the ROI the phonological features are found by the following steps (see Figure 1):

- Convolve with 16 × 16 pixel low-pass filter    (to remove noise)
- Extract a vertical intensity profile           (mouth height)
- Extract a horizontal intensity profile         (mouth width)
- Locate and label intensity peaks and valleys   (candidates for teeth, tongue)
- Calculate interframe peak motion               (speed estimates)

Video preprocessing tasks, including temporal averaging, are usually complicated because they require identifying corresponding pixels across frames. We circumvent this pixel correspondence problem by matching labeled features (such as intensity extrema — peaks and valleys) on successive frames.

## 2.1   FEATURES

The seventeen video features which serve as input to our recognizer are:
- Horizontal separation between the left and right mouth corners
- Vertical separation between the top and bottom lips

For each of the three vertically aligned positions:
- Vertical position:                    $P_v$
- Intensity value:                       $I$
- Change in intensity versus time:   $\Delta I/\Delta t$

For both of the mouth corner positions:
- Horizontal position:                   $P_h$
- Intensity value:                        $I$
- Change in intensity versus time:   $\Delta I/\Delta t$

For each speaker, each feature was scaled have a zero mean and unit standard deviation.

## 3   DATA COLLECTION AND NETWORK TRAINING

We trained the modified time delay neural network (Waibel, 1989) shown in Figure 2 on both the video and acoustic data. (See Stork, Wolff and Levine (1992) for a complete description of the architecture.) For the video only (VO) network, the input layer consists of 24 samples of each of the 17 features, corresponding to roughly 0.8 seconds. Each (sigmoidal) hidden unit received signals from a receptive field of 17 features for five consecutive frames. Each of the different hidden units (there were 3 for the results reported below) is replicated to cover the entire input space with overlapping receptive fields. The next layer consisted of 5 rows of x-units (one row for each possible utterance), with exponential transfer functions. They received inputs from the hidden units for 11 consecutive frames, thus they indirectly received input from a total of 18 input frames corresponding to roughly 0.6 seconds. The activities of the x-units encode the likelihood that a given letter occurs in that interval. The final layer consists of five p-units (probability units), which encode the relative probabilities of the presence of each of the possible utterances across the entire input window. Each p-unit sums the entire row of corresponding x-units, normalized by the sum over all x-units. (Note that "weights" from the x-units to the p-units are fixed.)

The acoustic only (AO) network shared the same architecture, except that the input consisted of 100 frames (1 second) of 14 mel scale coefficients each, and the x-units received fan in from 25 consecutive hidden units.

In the TDNN architecture, weights are shared, i.e., the pattern of input-to-hidden weights is forced to be the same at each interval. Thus the total number of independent weights in this VO network is 428, and 593 for the AO network.

These networks were trained using Backpropagation to minimize the Kullback-Leibler distance (cross-entropy) between the targets and outputs,

$$E \equiv D(t \parallel p) = \sum_i t_i \ln(\frac{t_i}{p_i}). \qquad (1)$$

Here the target probability is 1 for the target category, and 0 for all other categories. In this case Equation 1 simplifies to $E = -\ln(p_c)$ where $c$ is the correct category.

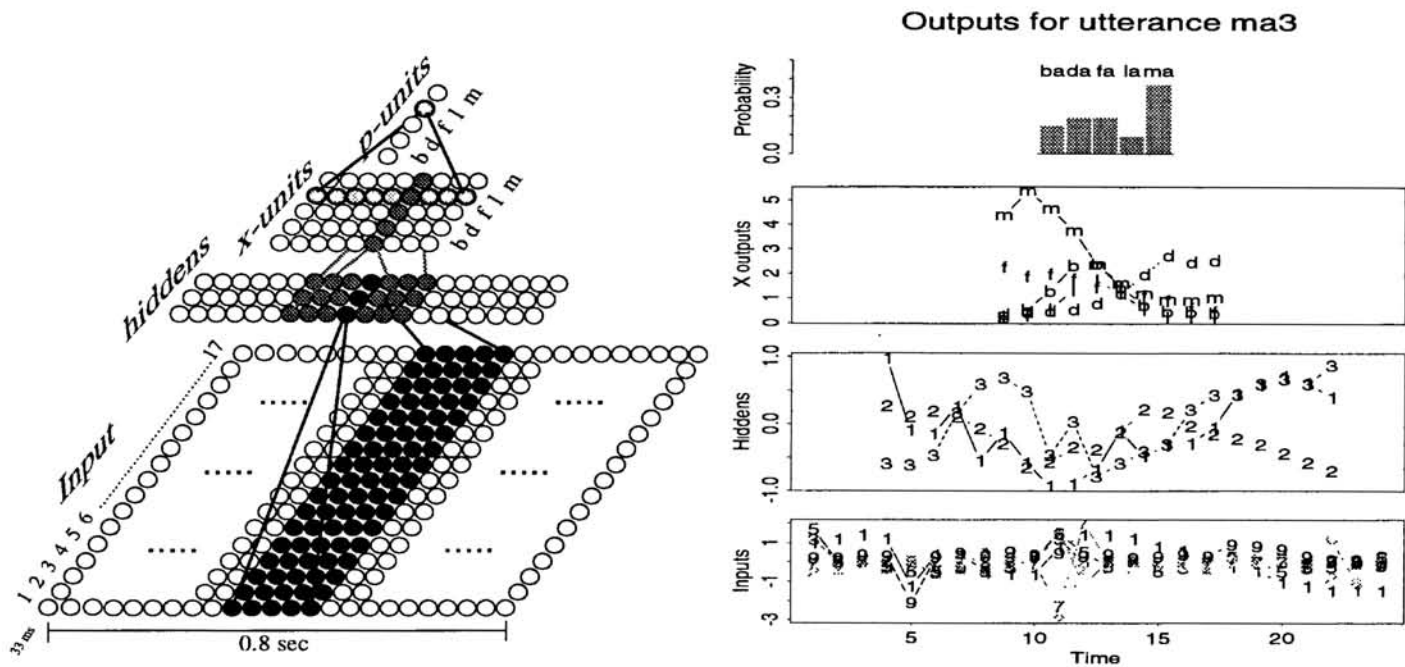

Figure 2: Modified time delay neural network architecture (left) and unit activities for a particular pattern (right). The output probabilities are calculated by integrating over the entire input window and normalizing across categories. Note the temporal segmentation which naturally occurs in the layer of X-units.

## 3.1 SENSORY INTEGRATION

Given the output probability distributions of the two networks, we combine them assuming conditional independence and using Bayes rule to obtain:

$$P(c_i|A, V) = \frac{P(A|c_i)P(V|c_i)P(c_i)}{\sum_{j=1}^{N} P(A|c_j)P(V|c_j)P(c_j)} \quad (2)$$

That is, the joint probability of the utterance belonging to category $c_i$ is just the normalized product of the outputs for category $c_i$ of each network.

This "independent opinion pooling"(Berger, 1985) offers several advantages over other methods for combining the modalities. First, it is optimal if the two signals really are conditionally independent, which appears to be the case for our data. (Proving that two signals are not conditionally independent is difficult.) Moreover, Massaro and Cohen (1983) have shown that human recognition performance is consistent with the independence assumption. A second advantage is simplicity. The combination adds no extra parameters beyond those used to model each signal, thus generalization performance should be good. Furthermore, the independent recognizers can be developed and trained separately, the only requirement is that they both output probability estimations.

A third advantage is that this system automatically compensates for noise and assigns more importance to the network which is most sure of its classification. For example, if the video data were very noisy (or missing), the video network would

judge all utterances equally likely. In this case the video contribution would cancel out, and the final output probabilities would be determined solely by the audio network. Bregler et al. (1993) attempt to compensate for the variance between channels by using the entropy of the output of the individual networks as a weighting on their contribution to the final outputs. Their ad hoc method suffers several drawbacks. For example, it does not distinguish the case where a one category is highly likely and the rest equiprobable, from the case where several categories are moderately likely.

A final advantage of Eq. 2 is that it does not require synchrony of the acoustic and visual features. The registration between the two signals could be off substantially (as long as the same utterance is present in the input to both networks). On the contrary, methods which attempt to detect cross-modal features would be very sensitive to the relative timing of the two signals.

## 4   RESULTS

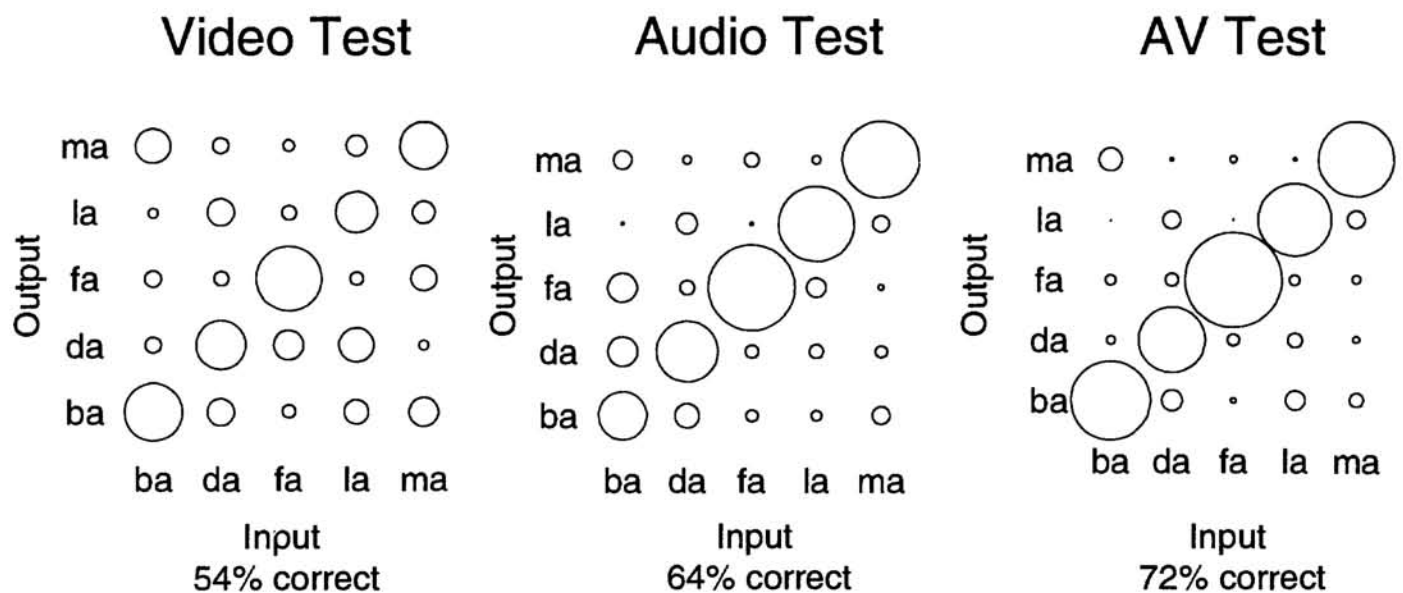

Figure 3: Confusion matrices for the video only (**VO**), acoustic only (**AO**), and the **AV** networks. Each vertical column is labeled by the spoken CV pair presented as input; each horizontal row represents the output by the network. The radius of each disk in the array is proportional to the output probability given an input letter. The recognition accuracy (measured as a percentage of novel test patterns properly classified by maximum network output) is shown.

The video and audio networks were trained separately on several different consonants in the same vowel context (/ba/, /da/, /fa/, /la/, /ma/) recorded from several different speakers. (For the results reported below, there were 10 speakers, repeating each of 5 CV pairs 5 times. Four of these were used for training, and one for testing generalization.)

For the video only networks, the correct classification (using the Max decision rule) on unseen data is typically 40-60%. As expected, the audio networks perform better with classification rates in the 50-70% range on these small sets of similar utterances.

Figure 3 shows the confusion matrices for the network outputs. We see that for the video only network the confusion matrix is fairly diagonal, indicating generally good performance. However the video network does tend to confuse utterances such as /ba/ ↔ /ma/. The audio network generally makes fewer errors, but confuses other utterances, such as /ba/ ↔ /da/.

The performance for the combined outputs (the AV network) is much better than either of the individual networks, achieving classification rates above 70%. (In previous work with only 4 speakers, classifications rates of up to 95% have been achieved.) We also see a strongly diagonal confusion matrix for the AV network, indicating that complementary nature of the the confusions made by the individual networks.

## 5    RELATIONSHIP TO HUMAN PERCEPTION

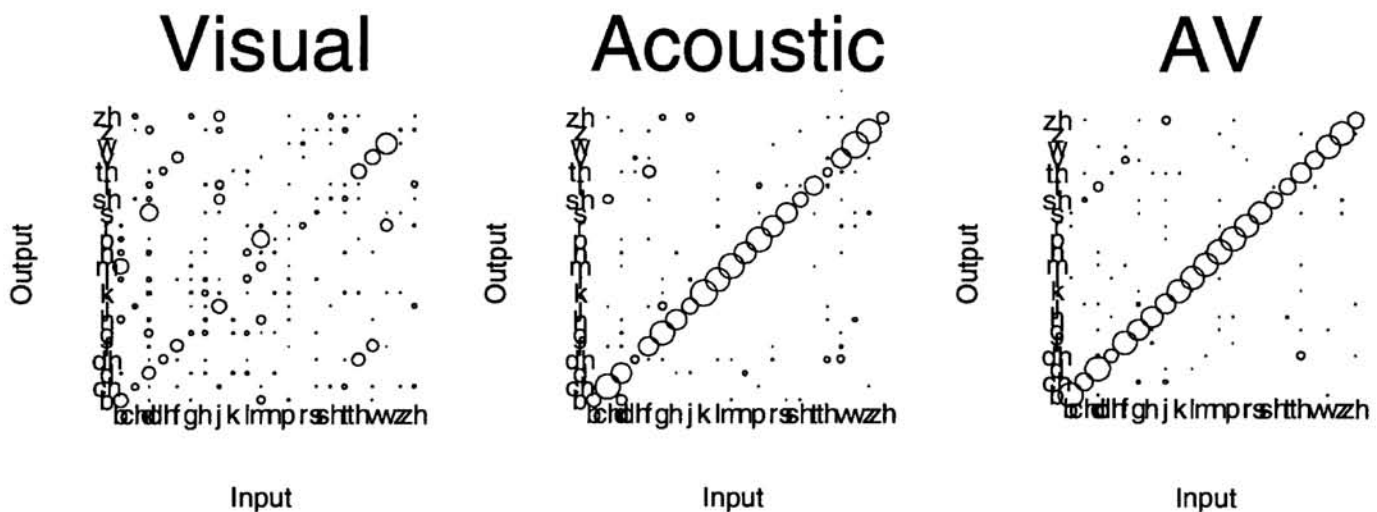

Figure 4: Confusion matrices from human recognition performance for video only, acoustic only, and combined speech for CV pairs (Massaro et al., 1993).

Interestingly, our results are qualitatively similar to findings in human perception. Massaro et al. (1993) presented Visual only, Acoustic only, and combined speech to subjects and collected response probabilities. As can be seen in the confusion matrices of Figure 4, subjects are not so bad at lipreading. The Visual only confusion matrix shows a strong diagonal component, though confusions such as /ma/ ↔ /ba/ are common. Performance on acoustic speech is better, of course, but there are still confusions such as /ba/ ↔ /da/. Combined speech yields even better recognition performance, eliminating most confusions. In fact, Massaro et al. found that the response probabilities of combined speech are accurately predicted by the product of the two single mode response probabilities. Massaro uses this and other evidence to argue quite convincingly that humans treat acoustic and visual speech channels independently, combining them only at a rather late stage of processing.

# 6 CONCLUSIONS AND FUTURE WORK

The video pre-processing presented here represents a first pass at reducing the amount of visual data to a manageable level in order to enable on-line processing. Our results indicate that even these straightforward, computationally tractable methods can significantly enhance speech recognition. Future efforts will concentrate on refining the pre-processing to capture more information, such as rounding and f-tuck, and testing the efficacy of our recognition system on larger datasets. The complementary nature of the acoustic and visual signals lead us to believe that a further refined speech reading system will significantly improve the state-of-the-art acoustic recognizers, especially in noisy environments.

## References

J. O. Berger. (1985) *Statistical decision theory and Bayesian analysis (2nd ed.)*. 272-275, New York: Springer-Verlag.

C. Bregler, S. Manke, H. Hild & A. Waibel. (1993) Bimodal Sensor Integration on the example of "Speech-Reading". Proc. ICNN-93, Vol. II 667-677.

C. L. De Filippo & D. G. Sims (eds.), (1988) *New Reflections on Speechreading* (Special issue of *The Volta Review*). **90**(5).

B. Dodd & R. Campbell (eds.). (1987) *Hearing by Eye: The Psychology of Lip-reading.* Hillsdale, NJ: Lawrence Erlbaum Press.

K. P. Green & J. L. Miller. (1985) On the role of visual rate information in phonetic perception. *Perception and Psychophysics* **38**, 269-276.

D. W. Massaro & M. M. Cohen (1983) Evaluation and integration of visual and auditory information in speech perception *J. Exp. Psych: Human Perception and Performance* **9**, 753-771.

D. W. Massaro, M. M. Cohen & A. T. Gesi (1993). Long-term training, transfer, and retention in learning to lipread. *Perception & Psychophysics*, 53, 549-562.

A. Pentland & K. Mase (1989) Lip reading: Automatic visual recognition of spoken words. *Proc. Image Understanding and Machine Vision, Optical Society of America*, June 12-14.

E. D. Petajan, B. Bischoff & D. Bodoff. (1988) An improved automatic lipreading system to enhance speech recognition. *ACM SIGCHI-88*, 19-25.

D. Sanders & S. Goodrich. (1971) The relative contribution of visual and auditory components of speech to speech intelligibility as a function of three conditions of frequency distortion. *J. Speech and Hearing Research* **14**, 154-159.

D. G. Stork, G. Wolff & E. Levine. (1992) Neural network lipreading system for improved speech recognition. Proc. IJCNN-92, Vol. II 285-295.

A. Waibel. (1989) Modular construction of time-delay neural networks for speech recognition. *Neural Computation* **1**, 39-46.

B. P. Yuhas, M. H. Goldstein, Jr., T. J. Sejnowski & R. E. Jenkins. (1988) Neural network models of sensory integration for improved vowel recognition. *Proc. IEEE* **78**(10), 1658-1668.
